# Orientational and geometric determinants of place and head-direction

**Neil Burgess & Tom Hartley**
Institute of Cognitive Neuroscience & Department of Anatomy, UCL
17 Queen Square, London WC1N 3AR, UK
*n.burgess@ucl.ac.uk. t.hartley@ucl.ac.uk*

## Abstract

We present a model of the firing of place and head-direction cells in rat hippocampus. The model can predict the response of individual cells and populations to parametric manipulations of both geometric (e.g. O'Keefe & Burgess, 1996) and orientational (Fenton et al., 2000a) cues, extending a previous geometric model (Hartley et al., 2000). It provides a functional description of how these cells' spatial responses are derived from the rat's environment and makes easily testable quantitative predictions. Consideration of the phenomenon of remapping (Muller & Kubie, 1987; Bostock et al., 1991) indicates that the model may also be consistent with non-parametric changes in firing, and provides constraints for its future development.

## 1   Introduction

'Place cells' recorded in the hippocampus of freely moving rats encode the rat's current location (O'Keefe & Dostrovsky, 1971; Wilson & McNaughton, 1993). In open environments a place cell will fire whenever the rat enters a specific portion of the environment (the 'place field'), independent of the rat's orientation (Muller et al., 1994). This location-specific firing appears to be present on the rat's first visit to an environment (e.g. Hill, 1978), and does not depend on the presence of local cues such as odors on the floor or walls. The complementary pattern of firing has also been found in related brain areas: 'head-direction cells' that fire whenever the rat faces in a particular direction independent of its location (Taube et al., 1990). Experiments involving consistent rotation of cues at or beyond the edge of the environment (referred to as 'distal' cues) produce rotation of the entire place (O'Keefe & Speakman, 1987; Muller et al., 1987) or head-direction (Taube et al., 1990) cell representation. Rotating cues within the environment does not produce this effect (Cressant et al., 1997). Here we suggest a predicitive model of the mechanisms underlying these spatial responses.

## 2   Geometric influences given consistent orientation

Given a stable directional reference (e.g. stable distal cues across trials), fields are determined by inputs tuned to detect extended obstacles or boundaries at particular

bearings. That is, they respond whenever a boundary or obstacle occurs at a given distance along a given allocentric direction, independent of the rat's orientation. These inputs are referred to below as putative 'boundary vector cells' (BVCs). The functional form of these inputs has been estimated by recording from the same place cell in several environments of differing geometry within the same set of distal orientation cues (O'Keefe & Burgess, 1996; Hartley et al., 2000). That is, for a BVC $i$ tuned to a boundary at distance $d_i$ and bearing $\phi_i$ relative to the rat, the response to a boundary segment at distance $r$ and bearing $\theta$, subtending an angle $\delta\theta$ at the rat, is given by:

$$\delta f_i = g_i(r, \theta)\delta\theta,$$

$$g_i(r, \theta) \propto \frac{\exp[-(r - d_i)^2/2\sigma_{rad}^2(d_i)]}{\sqrt{2\pi\sigma_{rad}^2(d_i)}} \times \frac{\exp[-(\theta - \phi_i)^2/2\sigma_{ang}^2]}{\sqrt{2\pi\sigma_{ang}^2}} \qquad (1)$$

where the angular width $\sigma_{ang}$ is a constant but the radial width $\sigma_{rad} = \sigma_0(1 + d_i/\beta)$ so that the width of tuning to distance increases with the distance of peak response $d_i$. Constants $\sigma_0$ and $\beta$ determine width at zero distance and its rate of increase with distance. The firing rate of BVC $i$, when the rat is at a location $\vec{x}$, is found by integrating $\delta f_i$ over $\theta$ (this is done numerically as the distance $r$ to the nearest boundary in direction $\theta$ is a function of $\vec{x}$, $\theta$ and the geometry of the environment). A place cell's firing rate $F(\vec{x})$ is then simply the thresholded linear sum of the firing rates of the $n$ BVCs connected to it, i.e.

$$F(\vec{x}) = A\Theta\left(\sum_{i=1}^{n}\left(\int_0^{2\pi} g_i(r(\vec{x}), \theta)d\theta\right) - T\right)$$

where $\Theta(x)$ is the Heaviside function ($\Theta(x) = x$ if $x > 0$; $\Theta(x) = 0$ otherwise). All simulations have $\beta = 183cm$, $\sigma_0 = 12.2cm$, $\sigma_{rad} = 0.2rad$, while the threshold $T$ can vary between simulations (e.g. between Figs. 1 and 2) but not between cells, and $A$ is an arbitrary constant as absolute firing rates are not shown.

Thus, in this model, a place cell's response is simply determined by the parameters $d_i$ and $\phi_i$ chosen for the set of BVCs connected to it. Assuming a random selection of BVCs for each place cell, and a single value for $T$, the model provides a good fit to the characteristics of populations of place fields across different environments, such as the distribution of firing rates and field shapes and sizes. Inputs can also be chosen so as to fit a given place field so that its behavior in a new environment of different shape can be predicted. See Hartley et al. (2000) and Fig. 1.

Like other models relying on the bearing to a landmark (Redish & Touretsky, 1996; McNaughton et al., 1996), the basic geometrical model assumes an accurate directional reference, but does not state how this depends on the sensory input. Note that, as such, this model already captures effects of consistent rotation of orientation cues around an environment as a reorientation of the directional reference frame that in turn affects the directions along which BVCs are tuned to respond. Indeed, the effect of consistent rotation of orientation cues about a environment of fixed geometry is identical to the rotation of the environment within a fixed directional reference frame, and can be modelled in this way (see e.g. the square and diamond in Figs. 1b,c).

## 3   Model of geometric and orientation influences

Models of head direction (Skaggs et al., 1995; Zhang, 1996) indicate how orientation might be derived. Internal inputs (e.g. vestibular or proprioceptive) maintain a consistent representation of heading within a ring of head-direction cells arranged

to form a continuous attractor. Correlational learning of associations from visual inputs to head direction cells then allows the representation of head direction to be maintained in synch with the external world. These models account for the preferential influence of large cues at a stable bearing (i.e. at or beyond the edge of the environment), and effects of instability caused by continual movement of cues or disorientation of the rat. They also allow orientation to be maintained in the face of cue removal, unless all cues are removed in which case orientation is wholly reliant on internal inputs and will drift over time. In this paper we take a step towards providing a quantitative model for the combined influences of orientation cues and boundaries on the firing of place and head direction cells. Such a model should be able to predict the behaviour of these cells under arbitrary environmental manipulations, bearing in mind that some (extended) objects may be both orientation cues and boundaries.

We focus on a series of experiments regarding inconsistent rotations of two extended cue cards (one white, one black) around the perimeter of a cylinder in the absence of any other orientation cues (Fenton et al., 2000a). Each of these cards controls the orientation of the set of place fields when rotated together or alone (after removal of the other cue). When both are rotated inconsistently, place fields are displaced in a non-uniform manner, with the displacement of a field being a function of its location within the environment. These findings cannot be explained by a simple rotation of the reference frame. Fig. 2A shows how place fields are displaced following counter rotation of the two cue cards. Since the cue cards are orientation cues and also walls of the environment, explaining these data within the current framework requires two separate considerations: i) how the movement of the cards affects the BVC's directional reference frame, and ii) how the movement of the cards, acting as boundaries, directly affects the BVCs.

We make the following assumptions:

1. The influence of a distal visual cue on the directional reference system is proportional to its proximity to the rat.

2. In the continued presence of color (or contrast) variation along a boundary to which a BVC responds, the BVC will become modulated by color: responding preferentially to, say, a white section of wall rather than the adjacent grey wall. In the absence of such variation it will revert to its unmodulated response.

We note that assumption 1) is consistent with most implementations of the head direction model discussed above, in that the influence of an extended distal cue will increase with the angle subtended by it at the rat. We also note that assumption 2) implies the presence of synaptic learning (something not required by the rest of the model), albeit outside of the hippocampus.

To avoid having to simulate enough random selections of BVCs to produce place fields at all locations within the environment and with all combinations of distance, bearing and color preferences, the model must be further simplified. To model the effect of cue manipulation on a place field in a location from which there are two cue cards at distances $D_i$ and bearings $\Phi_i$, we simulate a place cell for that location which receives inputs from two BVCs tuned to the distances $D_i$ and bearings $\Phi_i$, and to the most common color of boundary segments to which it respondes (across all positions of the rat). That is, $d_i = D_i$ and $\phi_i = \Phi_i$ in equation 1. For each location in the environment, we compute the shape of the place field formed by the thresholded sum of these BVCs, before and after the cue card manipulation. This simplification is broadly representative of the qualitative effect of the manipulation

on the locations of place fields[1].

How does this model campare to the Fenton et al. data? First we note that (due to assumption 1) each cue card can control the overall orientation of the place and head-direction representations. Similarly removing a cue card will have little effect, save for a slight rotation and/or transverse spreading of the BVC that responds to it (as it is no longer constrained by the color boundary, see assumption 2). When the cues are rotated inconsistently, the firing fields of the BVCs move relative to each other. The net effect of this on place fields and their centroids (Fig. 2B) compares well with the data (Fig. 2A) and is composed of two separate effects. First, the rotation of the cues produces a non-uniform distortion of the head direction system. The extent of rotation depends on the location of the animal relative to the cues as the closer a cue the more it affects the directional reference at that location (assumption 1) see Fig. 2C (ii). This distortion of the directional reference frame affects the orientations to which the boundary vector cells are tuned, and thus affects the location of place fields in an approximately rotational manner see Fig. 2C (iii). Second, the movement of the cue cards directly affects the firing fields of the BVCs due to their color preferrence. This 'translational' effect is shown in Fig. 2C (iv). Note that neither translational nor rotational effects alone are sufficient to explain the observed data. Fenton et al. (2000b) also make a distinction between translation and rotation in producing a phenomenological model of their data. However, as such, their model does not provide a mechanistic account at the level of cells, is specific to the cue-card manipulation they made and so does not make any prediction for head-direction cells or place cells in other experiments.

## 4  Non-parametric changes: 'remapping'

Our model considers the pattern of firing of place cells when the rat is put into an environment of different shape, or when two very familiar landmarks are moved or removed. In these situations changes to patterns of firing tend to be parametric, and the model aims to capture the parametric relationships between firing pattern and environmental manipulation. However we note that, after several days or weeks of experience, the place cell representations of two environments of different shape gradually diverge (Lever et al., 2002), such that the final representations can be said to have 'remapped' (Muller and Kubie, 1987). After 'remapping' a given cell might fire in only one of the environments, or might fire in both but in unrelated locations. Additionally, changing the color of the cue card in a grey cylinder from white to black can cause more rapid remapping such that the effect on the first day is probably best described as a slight rotation, with remapping occurring by the second day (Bostock et al., 1991). Note that simply removing the cue card just causes the overall orientation of the place field representation to drift.

Could the current model be extended to begin to understand these apparently non-parametric changes? The effects of replacing the cue card with a novel one are consistent with assumption 2 and the extra-hippocampal learning it implies: BVCs initially respond to the new color as they would upon removal of the cue card, with

the slight rotation or spreading of the firing field noted above. Over time in the presence of the new color, the color modulation of BVCs sharpens such that those previously responding to white or grey no longer respond to black, while new BVCs that do respond to black begin to fire. Thus the original place fields (particularly those nearest the card and so most reliant on BVCs from that direction) will tend to fall below threshold, unless receiving a connection from a newly active black-sensitive BVC, in which case the field will change location. Equally, some previously silent place cells will become active due to input from a newly-active black-sensitive BVC. By contrast, the slow shape-dependent remapping would appear to require some additional mechanism. This may be related to the evidence of shorter-term learning of associations between place cells (Mehta et al., 1997) or the NMDA-dependent stability of place fields (Kentros et al., 1998) or postulated processes of learned orthogonalisation of hippocampal representations (Marr, 1971; McClelland et al., 1995; Treves & Rolls, 1992; Fuhs & Touretzky, 2000; Kali & Dayan, 2000).

## 5   Conclusion

The model we have presented is consistent with a large body of detailed data on the effects of parametric environmental manipulations on place and head-direction cells. More importantly, it is a *predictive* model at the level of individual cells. Fig. 2C (ii) shows the prediction resulting from assumption 1) regarding the effect of the inconsistent cue card manipulation on head-direction cells. We note that there is an alternative to this location-dependent warping of head direction responses: a direction-dependent warping such that responses to north directions are tilted northwestwards while responses to south directions are tilted southwestwards. This would correspond to the alternative assumption that the influence of a distal visual cue on a head direction cell is proportional to the similarity of the average direction of the cue from the rat and the preferred direction of the cell. We chose to simulate the former (assumption 1) as this is consistent with current head-direction models in keeping the angular separation of preferred directions constant (but rotating all of them together as a function of the proximity of the rat to one or other cue card). The alternative assumption breaks this constancy, but would produce roughly equivalent results for place cell firing. Thus, on the basis of the Fenton et al. experiment on place cells we must predict one or other of the two effects on head-direction, or some combination of both. Beyond this, the model can predict the effect of essentially arbitrary parametric movements of cues and boundaries on place and head-direction cells over the short term. It also appears to be at least consistent with the non-parametric 'remapping' changes induced by color changes. Whether or not it can also predict the statistics of remapping over longer timescales in response to purely geometric changes is a question for future work.

**Acknowledgements:** We thank John O'Keefe, Colin Lever and Bob Muller for many useful discussions.

## 6   References

Bostock E, Muller RU, Kubie JL (1991) Experience-dependent modifications of hippocampal place cell firing Hippocampus 1, 193-206.

Cressant A, Muller RU, Poucet B (1997) Failure of centrally placed objects to control the firing fields of hippocampal place cells. J. Neurosci. 17, 2531-2542.

Fenton AA, Csizmadia G, & Muller RU (2000a). Conjoint control of hippocampal place cell firing by two visual stimuli. I. The effects of moving the stimuli on firing field positions. J. Gen. Physiol, 116, 191-209.

Fenton AA, Csizmadia G, & Muller RU (2000b). Conjoint control of hippocampal place cell firing by two visual stimuli. Ii. A vector-field theory that predicts modifications of the representation of the environment. J. Gen. Physiol, 116, 211-221.

Fuhs MC, Touretzky DS (2000) Synaptic learning models of map separation in the hippocampus. Neurocomputing, 32:379-384.

Hartley T, Burgess N, Lever C, Cacucci F, O'Keefe J (2000) Modeling place fields in terms of the cortical inputs to the hippocampus. Hippocampus, 10, 369-379.

Hill AJ (1978) First occurrence of hippocampal spatial firing in a new environment. Exp. Neurol 62, 282-297.

Káli S, Dayan P (2000) The Involvement of Recurrent Connections in Area CA3 in Establishing the Properties of Place Fields: A Model. J. Neurosci. 20, 7463-7477.

Kentros C, Hargreaves E, Hawkins RD, Kandel ER, Shapiro M, Muller RU (1998) Abolition of long-term stability of new hippocampal place cell maps by NMDA receptor blockade. Science, 280, 2121-2126.

McNaughton BL, Knierim JJ, Wilson MA (1994) 'Vector encoding and the vestibular foundations of spatial cognition: a neurophysiological and computational hypothesis', In *The Cognitive Neurosciences*, (ed. Gazzaniga, M.) 585-596 (MIT Press, Boston, 1994).

Lever CL, Wills T, Cacucci F, Burgess N, O'Keefe J (2002) Long-term plasticity in the hippocampal place cell representation of environmental geometry. Nature, in press.

Marr D (1971) Simple memory: a theory for archicortex. Phil. Trans. Roy. Soc. Lond B 262, 23-81.

McClelland JL, McNaughton BL, O'Reilly RC (1995) Why there are complementary learning-systems in the hippocampus and neocortex - insights from the successes and failures of connectionist models of learning and memory. Psychological Review 102, 419-457.

Mehta MR, Barnes CA, McNaughton BL (1997) Experience-dependent, asymmetric expansion of hippocampal place fields. Proc. Nat. Acad. Sci. 94, 8918-8921.

Muller RU, Bostock E, Taube JS, Kubie JL (1994) On the directional firing properties of hippocampal place cells. J. Neurosci. 14 7235-7251.

Muller RU, Kubie JL (1987) The effects of changes in the environment on the spatial firing of hippocampal complex-spike cells. J. Neurosci 7, 1951-1968.

Muller RU, Kubie JL, Ranck JB (1987) Spatial firing patterns of hippocampal complex-spike cells in a fixed environment. J. Neurosci., 7, 1935-1950.

O'Keefe J, Burgess N (1996) Geometric Determinants of the Place Fields of Hippocampal Neurones. Nature 381, 425-428.

O'Keefe J, Dostrovsky J (1971) The hippocampus as a spatial map: preliminary evidence from unit activity in the freely moving rat. Brain Res 34, 171-175.

O'Keefe J, Speakman A (1987) Single unit activity in the rat hippocampus during a spatial memory task. Exp. Brain Res 68, 1-27.

Redish AD, Touretzky DS (1996) Modeling interactions of the rat's place and head direction systems *Advances in Neural Information Processing Systems, 8.* D Touretzky, MC Mozer, ME Hasselmo (eds) pp. 61-67. MIT Press, Cambridge MA.

Skaggs WE, Knierim JJ, Kudrimoti HS, McNaughton BL (1995) 'A model of the neural basis of the rat's sense of direction' *Advances in Neural Information Processing Systems, 7.* G Tesauro, D Touretzky & TK Leen (eds) pp. 51-58. MIT Press, Cambridge MA.

Taube JS, Muller RU, Ranck JB (1990) Head-direction cells recorded from the postsubiculum in freely moving rats. I. Description and quantitative analysis. J. Neurosci 10, 420-435.

Treves A, Rolls ET (1992) Computational constraints suggest the need for two distinct input systems to the hippocampal CA3 network. Hippocampus 2, 189-200.

Wilson MA, McNaughton BL (1993) Dynamics of the hippocampal ensemble code for space. Science 261, 1055-1058.

Zhang K (1996) Representation of spatial orientation by the intrinsic dynamics of the head-direction cell ensemble: a theory. J.Neurosci., 16, 2112-2126.

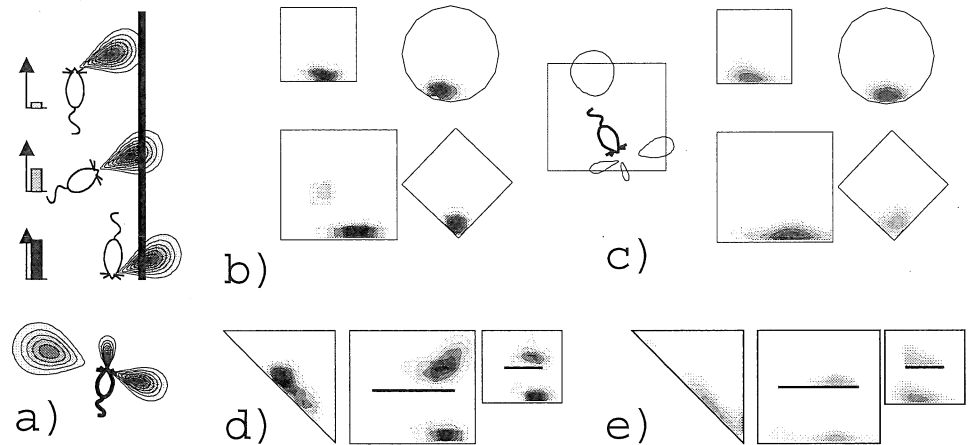

Figure 1: Model of the geometrical influence on place fields (adapted from Hartley et al., 2000), assuming a stable directional reference frame. Place fields are composed from thresholded linear sums of the firing rates of 'boundary vector cells' (BVCs). a) Above: Each BVC has a Gaussian tuned response to the presence of a boundary at a given distance and bearing from the rat (independent of its orientation). Below: The sharpness of tuning of a BVC decreases as the distance to which it is tuned increases. The only free parameters of a BVC are the distance and direction of peak response. b) Place fields recorded from the same cell in four environments of different shape or orientation relative to distal cues. c) Simulation of the place fields in b) by the best fitting set of 4 BVCs constrained to be in orthogonal directions (BVCs shown on the left, simulated fields on the right). The simulated cell can now be used to predict firing in novel situations. Real and predicted data from three novel environments are shown in d) and e) respectively, showing good qualitative agreement.

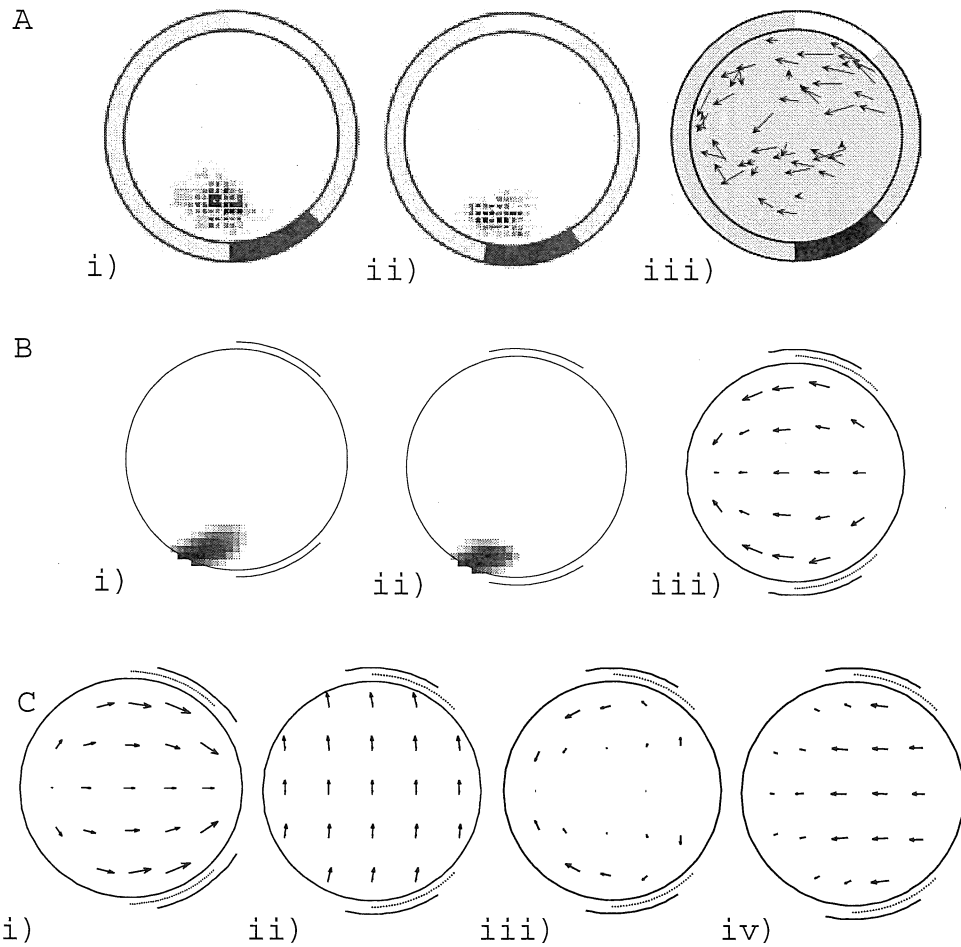

Figure 2: Changes to place fields in a cylinder following inconsistent rotation of two cue cards. A) Experimental data shown in a birds-eye view of the cyclinder including the black and white cue-cards (adapted from Fenton et al., 2000a). i) A place field with the cue cards in the 'standard' condition (used throughout training). ii) The place field after inconsistent rotation of each cue card by 12.5° further apart ('apart' condition). iii) The movement of the centroid of place field from the standard condition (tail of arrow) to the apart condition (head of arrow). B) Simulation of 21 place fields in the cyclinder in standard and apart conditions. Cue card locations are indicated by a black line (initial card positions are indicated by a dotted line to illustrate changes from one condition to another). i) and ii) show the place field nearest in location to that shown in A) in standard and apart conditions. iii) shows the movement of the centroids of simulated place fields between standard and apart conditions. C) i) Simulation of the movement of place field centroids between the standard and 'together' conditions (cue cards rotated 12.5° closer together). ii) The distortion of the preferred direction of a head direction cell. Arrows show the preferred direction in the 'apart' condition, the preferred direction was 'up' in the standard condition. iii) the movement of place field centroids between the standard and apart condition due solely to the directional distortion shown in ii). iv) the movement of place field centroids due solely to the movement of the cue cards acting as distinct cues (without any directional distortion shown in ii). The net effect of fields iii) and iv) is that shown in B iii).

## Footnotes

[1]Simulations of place fields with a larger number of BVCs indicate similar field movements, but of reduced magnitude in locations far from the cue cards. However the good match between the simple model and the data (Figs. 2A,B) suggests that the cue cards do provide the majority of BVC input. This might be due to learned salience over the extensive training period, and to the learning process implied by assumption 2. Against this, place fields formed by more the two BVC inputs (e.g. the four BVCs in Fig. 1c) generally give a better match to field shape, especially in locations far from the two cue cards.
